# A Formulation for Minimax Probability Machine Regression

**Thomas Strohmann**
Department of Computer Science
University of Colorado, Boulder
*strohman@cs.colorado.edu*

**Gregory Z. Grudic**
Department of Computer Science
University of Colorado, Boulder
*grudic@cs.colorado.edu*

## Abstract

We formulate the regression problem as one of maximizing the minimum probability, symbolized by $\Omega$, that future predicted outputs of the regression model will be within some $\pm\varepsilon$ bound of the true regression function. Our formulation is unique in that we obtain a *direct* estimate of this lower probability bound $\Omega$. The proposed framework, minimax probability machine regression (MPMR), is based on the recently described minimax probability machine classification algorithm [Lanckriet *et al.*] and uses Mercer Kernels to obtain nonlinear regression models. MPMR is tested on both toy and real world data, verifying the accuracy of the $\Omega$ bound, and the efficacy of the regression models.

## 1 Introduction

The problem of constructing a regression model can be posed as maximizing the minimum probability of future predictions being within some bound of the true regression function. We refer to this regression framework as minimax probability machine regression (MPMR). For MPMR to be useful in practice, it must make minimal assumptions about the distributions underlying the true regression function, since accurate estimation of these distribution is prohibitive on anything but the most trivial regression problems. As with the minimax probability machine classification (MPMC) framework proposed in [1], we avoid the use of detailed distribution knowledge by obtaining a *worst case bound* on the probability that the regression model is within some $\varepsilon > 0$ of the true regression function. Our regression formulation closely follows the classification formulation in [1] by making use of the following theorem due to Isii [2] and extended by Bertsimas and Sethuraman [3]:

$$\sup_{E[\mathbf{z}]=\bar{\mathbf{z}},Cov[\mathbf{z}]=\Sigma_{\mathbf{z}}} Pr\{\mathbf{a}^T\mathbf{z} \geq b\} = \frac{1}{1+\omega^2}, \ \omega^2 = \inf_{\mathbf{a}^T\mathbf{z}\geq b}(\mathbf{z}-\bar{\mathbf{z}})^T\Sigma_{\mathbf{z}}^{-1}(\mathbf{z}-\bar{\mathbf{z}}) \quad (1)$$

where $\mathbf{a}$ and $b$ are constants, $\mathbf{z}$ is a random vector, and the supremum is taken over all distributions having mean $\bar{\mathbf{z}}$ and covariance matrix $\Sigma_{\mathbf{z}}$. This theorem assumes linear boundaries, however, as shown in [1], Mercer kernels can be used to obtain nonlinear versions of this theorem, giving one the ability to estimate upper and lower bounds on probability that points generated form any distribution having mean $\bar{\mathbf{z}}$ and covariance $\Sigma_{\mathbf{z}}$, will be on one side of a nonlinear boundary. In [1], this formulation is used to construct nonlinear classifiers (MPMC) that maximize the minimum probability of correct classification on future data.

In this paper we exploit the above theorem (**??**) for building nonlinear regression functions which maximize the minimum probability that the future predictions will be within an $\varepsilon$ to the true regression function. We propose to implement MPMR by using MPMC to construct a classifier that separates two sets of points: the first set is obtained by shifting all of the regression data $+\varepsilon$ along the dependent variable axis; and the second set is obtained by shifting all of the regression data $-\varepsilon$ along the dependent variable axis. The the separating surface (i.e. classification boundary) between these two classes corresponds to a regression surface, which we term the minimix probability machine regression model. The proposed MPMR formulation is unique because it *directly* computes a bound on the probability that the regression model is within $\pm\varepsilon$ of the true regression function (see Theorem 1 below).

The theoretical foundations of MPMR are formalized in Section 2. Experimental results on synthetic and real data are given in Section 3, verifying the accuracy of the minimax probability regression bound and the efficacy of the regression models. Proofs of the two theorems presented in this paper are given in the appendix. Matlab and C source code for generating MPMR models can be downloaded from *http://www.cs.colorado.edu/~grudic/software*.

## 2 Regression Model

We assume that learning data is generated from some unknown regression function $f : \Re^d \mapsto \Re$ that has the form:
$$y = f(\mathbf{x}) + \rho \tag{2}$$
where $\mathbf{x} \in \Re^d$ are generated according to some bounded distribution $\Lambda$, $y \in \Re$, $E[\rho] = 0$, $Var[\rho] = \sigma^2$, and $\sigma \in \Re$ is finite. We are given $N$ learning examples $\Gamma = \{(\mathbf{x}_1, y_1), ..., (\mathbf{x}_N, y_N)\}$, where $\forall i \in \{1, ..., N\}$, $\mathbf{x}_i = (x_{i1}, ..., x_{id}) \in \Re^d$ is generated from the distribution $\Lambda$, and $y_i \in \Re$. The goal of our formulation is two-fold: first we wish to use $\Gamma$ to construct an approximation $\hat{f}$ of $f$, such that, for *any* $\mathbf{x}$ generated from the distribution $\Lambda$, we can approximate $\hat{y}$ using
$$\hat{y} = \hat{f}(\mathbf{x}) \tag{3}$$
The second goal of our formulation is, for any $\varepsilon \in \Re$, $\varepsilon > 0$, estimate the bound on the minimum probability, symbolized by $\Omega$, that $\hat{f}(\mathbf{x})$ is within $\varepsilon$ of $y$ (define in (2)):
$$\Omega = \inf \Pr\{|\hat{y} - y| \le \varepsilon\} \tag{4}$$
Our proposed formulation of the regression problem is unique because we obtain direct estimates of $\Omega$. Therefore we can estimate the predictive power of a regression function by a bound on the minimum probability that we are within $\varepsilon$ of the true regression function. We refer to a regression function that directly estimates (4) as a mimimax probability machine regression (MPMR) model.

The proposed MPMR formulation is based on the kernel formulation for mimimax probability machine classification (MPMC) presented in [1]. Therefore, the MPMR model has the form:
$$\hat{y} = \hat{f}(\mathbf{x}) = \sum_{i=1}^{N} \beta_i K(\mathbf{x}_i, \mathbf{x}) + b \tag{5}$$
where, $K(\mathbf{x}_i, \mathbf{x}) = \varphi(\mathbf{x}_i)\varphi(\mathbf{x})$ is a kernel satisfying Mercer's Conditions, $\mathbf{x}_i$, $\forall i \in \{1, ..., N\}$, are obtained from the learning data $\Gamma$, and $\beta_i, b \in \Re$ are outputs of the MPMR learning algorithm.

### 2.1 Kernel Based MPM Classification

Before formalizing the MPMR algorithm for calculating $\beta_i$ and $b$ from the training data $\Gamma$, we first describe the MPMC formulation upon which it is based. In [1], the binary classification problem is posed as one of maximizing the probability of correctly classifying future

data. Specifically, two sets of points are considered, here symbolized by $\{\mathbf{u}_1, ..., \mathbf{u}_{N_u}\}$, where $\forall i \in \{1, ..., N_u\}, \mathbf{u}_i \in \Re^m$, belonging to the first class, and $\{\mathbf{v}_1, ..., \mathbf{v}_{N_v}\}$, where $\forall i \in \{1, ..., N_v\}, \mathbf{v}_i \in \Re^m$, belonging to the second class. The points $\mathbf{u}_i$ are assumed to be generated from a distribution that has mean $\overline{\mathbf{u}}$ and a covariance matrix $\Sigma_{\mathbf{u}}$, and correspondingly, the points $\mathbf{v}_i$ are assumed to be generated from a distribution that has mean $\overline{\mathbf{v}}$ and a covariance matrix $\Sigma_{\mathbf{v}}$. For the nonlinear kernel formulation, these points are mapped into a higher dimensional space $\varphi : \Re^m \mapsto \Re^h$ as follows: $\mathbf{u} \mapsto \varphi(\mathbf{u})$ with corresponding mean and covariance matrix $(\overline{\varphi(\mathbf{u})}, \Sigma_{\varphi(\mathbf{u})})$, and $\mathbf{v} \mapsto \varphi(\mathbf{v})$ with corresponding mean and covariance matrix $(\overline{\varphi(\mathbf{v})}, \Sigma_{\varphi(\mathbf{v})})$. The binary classifier derived in [1] has the form ($c = -1$ for the first class and $c = +1$ for the second):

$$c = \text{sign}\left[\sum_{i=1}^{N_u+N_v} \gamma_i K^c(\mathbf{z}_i, \mathbf{z}) + b_c\right] \tag{6}$$

where $K^c(\mathbf{z}_i, \mathbf{z}) = \varphi(\mathbf{z}_i)\varphi(\mathbf{z})$, $\mathbf{z}_i = \mathbf{u}_i$ for $i = 1, ..., N_u$, $\mathbf{z}_i = \mathbf{v}_{i-N_u}$ for $i = N_u + 1, ..., N_u + N_v$, and $\gamma = (\gamma_1, ..., \gamma_{N_u+N_v}), b_c$ obtained by solving the following optimization problem:

$$\min_{\gamma}\left\{\left\|\frac{\tilde{\mathbf{K}}_{\mathbf{u}}}{\sqrt{N_u}}\gamma\right\|_2 + \left\|\frac{\tilde{\mathbf{K}}_{\mathbf{v}}}{\sqrt{N_v}}\gamma\right\|_2\right\} s.t. \gamma^T\left(\tilde{\mathbf{k}}_{\mathbf{u}} - \tilde{\mathbf{k}}_{\mathbf{v}}\right) = 1 \tag{7}$$

where $\tilde{\mathbf{K}}_{\mathbf{u}} = \mathbf{K}_{\mathbf{u}} - \mathbf{1}_{N_u}\tilde{\mathbf{k}}_{\mathbf{u}}$; where $\tilde{\mathbf{K}}_{\mathbf{v}} = \mathbf{K}_{\mathbf{v}} - \mathbf{1}_{N_v}\tilde{\mathbf{k}}_{\mathbf{v}}$; where $\tilde{\mathbf{k}}_{\mathbf{v}}, \tilde{\mathbf{k}}_{\mathbf{u}} \in \Re^{N_u+N_v}$ defined as: $[\tilde{\mathbf{k}}_{\mathbf{v}}]_i = \frac{1}{N_v}\sum_{j=1}^{N_v} K^c(\mathbf{v}_j, \mathbf{z}_i)$ and $[\tilde{\mathbf{k}}_{\mathbf{u}}]_i = \frac{1}{N_u}\sum_{j=1}^{N_u} K^c(\mathbf{u}_j, \mathbf{z}_i)$; where $\mathbf{1}_k$ is a $k$ dimensional column vector of ones; where $\mathbf{K}_{\mathbf{u}}$ contains the first $N_u$ rows of the Gram matrix $\mathbf{K}$ (i.e. a square matrix consisting of the elements $\mathbf{K}_{ij} = K^c(\mathbf{z}_i, \mathbf{z}_j)$); and finally $\mathbf{K}_{\mathbf{v}}$ contains the last $N_v$ rows of the Gram matrix $\mathbf{K}$. Given that $\gamma$ solves the minimization problem in (7), $b_c$ can be calculated using:

$$b_c = \gamma^T\tilde{\mathbf{k}}_{\mathbf{u}} - \kappa\sqrt{\frac{1}{N_u}\gamma^T\tilde{\mathbf{K}}_{\mathbf{u}}^T\tilde{\mathbf{K}}_{\mathbf{u}}\gamma} = \gamma^T\tilde{\mathbf{k}}_{\mathbf{v}} + \kappa\sqrt{\frac{1}{N_v}\gamma^T\tilde{\mathbf{K}}_{\mathbf{v}}^T\tilde{\mathbf{K}}_{\mathbf{v}}\gamma} \tag{8}$$

where,

$$\kappa = \left(\sqrt{\frac{1}{N_u}\gamma^T\tilde{\mathbf{K}}_{\mathbf{u}}^T\tilde{\mathbf{K}}_{\mathbf{u}}\gamma} + \sqrt{\frac{1}{N_v}\gamma^T\tilde{\mathbf{K}}_{\mathbf{v}}^T\tilde{\mathbf{K}}_{\mathbf{v}}\gamma}\right)^{-1} \tag{9}$$

One significant advantage of this framework for binary classification is that, given perfect knowledge of the statistics $\overline{\mathbf{u}}, \Sigma_{\mathbf{u}}, \overline{\mathbf{v}}, \Sigma_{\mathbf{v}}$, the maximum probability of *incorrect* classification is bounded by $1 - \alpha$, where $\alpha$ can be directly calculated from $\kappa$ as follows:

$$\alpha = \frac{\kappa^2}{1 + \kappa^2} \tag{10}$$

This result is used below to formulate a lower bound on the probability that that the approximated regression function is within $\varepsilon$ of the true regression function.

## 2.2 Kernel Based MPM Regression

In order to use the above MPMC formulation for our proposed MPMR framework, we first take the original learning data $\Gamma$ and create two classes of points $\mathbf{u}_i \in \Re^{d+1}$ and $\mathbf{v}_i \in \Re^{d+1}$, for $i = 1, ..., N$, as follows:

$$\begin{aligned}\mathbf{u}_i &= (y_i + \varepsilon, x_{i1}, x_{i2}, ..., x_{id})\\ \mathbf{v}_i &= (y_i - \varepsilon, x_{i1}, x_{i2}, ..., x_{id})\end{aligned} \tag{11}$$

Given these two sets of points, we obtain $\gamma$ by minimizing equation (7). Then, from (6), the MPM classification boundary between points $\mathbf{u}_i$ and $\mathbf{v}_i$ is given by

$$\sum_{i=1}^{2N} \gamma_i K^c(\mathbf{z}_i, \mathbf{z}) + b_c = 0 \tag{12}$$

We interpret this classification boundary as a regression surface because it acts to separate points which are $\varepsilon$ above the $y$ values in the learning set $\Gamma$, and $\varepsilon$ below the $y$ values

in $\Gamma$. Furthermore, given any point $\mathbf{x} = (x_1, ..., x_d)$ generated from the distribution $\Lambda$, calculating $\hat{y}$ the regression model output (5), involves finding a $\hat{y}$ that solves equation (12), where $\mathbf{z} = (\hat{y}, x_1, ..., x_d)$, and, recalling from above, $\mathbf{z}_i = \mathbf{u}_i$ for $i = 1, ..., N$, $\mathbf{z}_i = \mathbf{v}_{i-N}$ for $i = N + 1, ..., 2N$ (note that $N_u = N_v = N$). If $K^c(\mathbf{z}_i, \mathbf{z})$ is nonlinear, solving (12) for $\hat{y}$ is in general a nonlinear single variable optimization problem, which can be solved using a root finding algorithm (for example the Newton-Raphson Method outlined in [4]). However, below we present a specific form of nonlinear $K^c(\mathbf{z}_i, \mathbf{z})$ that allows (12) to be solved analytically.

It is interesting to note that the above formulation of a regression model can be derived using any binary classification algorithm, and is not limited to the MPMC algorithm. Specifically, if a binary classifier is built to separate any two sets of points (11), then finding a crossing point $\hat{y}$ at where the classifier separates these classes for some input $\mathbf{x} = (x_1, ..., x_d)$, is equivalent to finding the output of the regression model for input $\mathbf{x} = (x_1, ..., x_d)$. It would be interesting to explore the efficacy of various classification algorithms for this type of regression model formulation. However, as formalized in Theorem 1 below, using the MPM framework gives us one clear advantage over other techniques. We now state the main result of this paper:

**Theorem 1:** *For any $\mathbf{x} = (x_1, ..., x_d)$ generated according to the distribution $\Lambda$, assume that there exists only one $\hat{y}$ that solves equation (12). Assume also perfect knowledge of the statistics $\overline{\mathbf{u}}$, $\Sigma_{\mathbf{u}}$, $\overline{\mathbf{v}}$, $\Sigma_{\mathbf{v}}$. Then, the minimum probability that $\hat{y}$ is within $\varepsilon$ of $y$ (as defined in (2)) is given by:*

$$\Omega = \inf \Pr\{|\hat{y} - y| \leq \varepsilon\} = \frac{\kappa^2}{1 + \kappa^2} \tag{13}$$

*where $\kappa$ is defined in (9).*

**Proof:** See Appendix.

Therefore, from the above theorem, the MPMC framework directly computes the lower bound on the probability that the regression model is within $\varepsilon$ of the function that generated the learning data $\Gamma$ (i.e. the true regression function). However, one key requirement of the theorem is perfect knowledge of the statistics $\overline{\mathbf{u}}$, $\Sigma_{\mathbf{u}}$, $\overline{\mathbf{v}}$, $\Sigma_{\mathbf{v}}$. In the actual implementation of MPMR, these statistics are estimated from $\Gamma$, and it is an open question (which we address in Section 3) as to how accurately $\Omega$ can be estimated from real data.

In order to avoid the use of nonlinear optimizations techniques to solve (12) for $\hat{y}$, we restrict the form of the kernel $K^c(\mathbf{z}_i, \mathbf{z})$ to the following:

$$K^c(\mathbf{z}_i, \mathbf{z}) = y_i'\hat{y} + K(\mathbf{x}_i, \mathbf{x}) \tag{14}$$

where $K(\mathbf{x}_i, \mathbf{x}) = \varphi(\mathbf{x}_i)\varphi(\mathbf{x})$ is a kernel satisfying Mercer's Conditions; where $\mathbf{z} = (\hat{y}, x_1, ..., x_d)$; where $\mathbf{z}_i = \mathbf{u}_i, y_i' = y_i + \epsilon$ for $i = 1, ..., N$; and where $\mathbf{z}_i = \mathbf{v}_{i-N}, y_{i-N}' = y_i - \epsilon$ for $i = N + 1, ..., 2N$. Given this restriction on $K^c(\mathbf{z}_i, \mathbf{z})$, we now state our final theorem which uses the following lemma:

**Lemma 1:**

$$\tilde{k}_u - \tilde{k}_v = 2\epsilon \mathbf{y}' \tag{15}$$

**Proof:** See Appendix.

**Theorem 2:** *Assume that (14) is true. Then all of the following are true:*
*Part 1: Equation (12) has an analytical solution as defined in (5), where*

$$\beta_i = -2\epsilon(\gamma_i + \gamma_{i+N})$$

$$b = -2\epsilon b_c$$

*Part 2:* $\tilde{\mathbf{K}}_{\mathbf{u}} = \tilde{\mathbf{K}}_{\mathbf{v}}$

Table 1: Results over 100 random trials for sinc data: mean squared errors and the standard deviation; MPTD$\varepsilon$: fraction of test points that are within $\epsilon = 0.2$ of $y$; predicted $\Omega$: predicted probability that the model is within $\varepsilon = 0.2$ of $y$.

| | | mean squared error | MPTD$\varepsilon$ | predicted $\Omega$ |
|---|---|---|---|---|
| $\sigma^2 = 0$ | mean (std) | 0.0 (0.0) | 1.0 (0.0) | 1.0 (0.0) |
| $\sigma^2 = 0.5$ | mean (std) | 0.0524 (0.0386) | 0.6888 (0.1133) | 0.1610 (0.0229) |
| $\sigma^2 = 1.0$ | mean (std) | 0.2592 (0.3118) | 0.3870 (0.1110) | 0.0463 (0.0071) |

*Part 3: The problem of finding an optimal $\gamma$ in (7) is reduced to solving the following linear least squares problem for* $\mathbf{t} \in \Re^{2N-1}$:

$$\min_{\mathbf{t}} \left\| \tilde{\mathbf{K}}_{\mathbf{u}} \left( \gamma_o + \mathbf{F}\mathbf{t} \right) \right\|_2$$

*where* $\gamma = \gamma_o + \mathbf{F}\mathbf{t}$, $\gamma_o = \left( \tilde{\mathbf{k}}_{\mathbf{u}} - \tilde{\mathbf{k}}_{\mathbf{v}} \right) / \left\| \tilde{\mathbf{k}}_{\mathbf{u}} - \tilde{\mathbf{k}}_{\mathbf{v}} \right\|^2$, *and* $\mathbf{F} \in \Re^{2N \times (2N-1)}$ *is an orthogonal matrix whose columns span the subspace of vectors orthogonal to* $\tilde{\mathbf{k}}_{\mathbf{u}} - \tilde{\mathbf{k}}_{\mathbf{v}}$.

**Proof:** See Appendix.

Therefore, Theorem 2 establishes that the MPMR formulation proposed in this paper has a closed form analytical solution, and its computational complexity is equivalent to solving a *linear* system of $2N - 1$ equations in $2N - 1$ unknowns.

## 3 Experimental Results

For complete implementation details of the MPMR algorithm used in the following experiments, see the Matlab and C source code available at *http://www.cs.colorado.edu/~grudic/software*.

**Toy Sinc Data:** Our toy example uses the noisy sinc function $y_i = sin(\pi x_i)/(\pi x_i) + \nu_i$ $i = 1, ..., N$, where $\nu_i$ is drawn from a Gaussian distribution with mean 0 and variance $\sigma^2$ [5]. We use a RBF kernel $K(\mathbf{a}, \mathbf{b}) = exp(-|\mathbf{a} - \mathbf{b}|^2)$ and $N = 100$ training examples. Figure 1 (a), (b), and (c), and Table 1 show the results for different variances $\sigma^2$ and a constant value of $\varepsilon = 0.2$. Figure 1 (d) and (e) illustrate how different tube sizes $0.05 \leq \varepsilon \leq 2$ affect the mean squared error (on 100 random test points), the predicted $\Omega$ and measured percentage of test data within $\varepsilon$ (here called MPTD$\varepsilon$) of the regression model. Each experiment consists of 100 random trials. The average mean squared error in (e) has a small deviation (0.0453) over all tested $\varepsilon$ and always was within the range 0.19 to 0.35. This indicates that the accuracy of the regression model is essentially independent from the choice of $\varepsilon$. Also note that the mean predicted $\Omega$ is a lower bound on the mean MPTD$\varepsilon$. The tightness of this lower bound varies for different amounts of noise (Table 1) and different choices of $\varepsilon$ (Figure 1 d).

**Boston Housing Data:** We test MPMR on the widely used Boston housing regression data available from the UCI repository. Following the experiments done in [5], we use the RBF kernel $K(\mathbf{a}, \mathbf{b}) = exp(-\|\mathbf{a} - \mathbf{b}\|/(2\sigma^2))$, where $(2\sigma^2)) = 0.3 \cdot d$ and $d = 13$ for this data set. No attempt was made to pick optimal values for $\sigma$ using cross validation. The Boston housing data contains 506 training examples, which we randomly divided into $N = 481$ training examples and 25 testing examples for each test run. 100 such random tests where run for each of $\varepsilon = 0.1, 1.0, 2.0, ..., 10.0$. Results are reported in Table 2 for 1) average mean squared errors and the standard deviation; 2) MPTD$\tilde{\varepsilon}$: fraction of test points that are within $\epsilon$ of $y$ and the standard deviation; 3) predicted $\Omega$: predicted probability that the model is within $\varepsilon$ of $y$ and standard deviation. We first note that the results compare favorably to those reported for other state of the art regression algorithms [5], even though

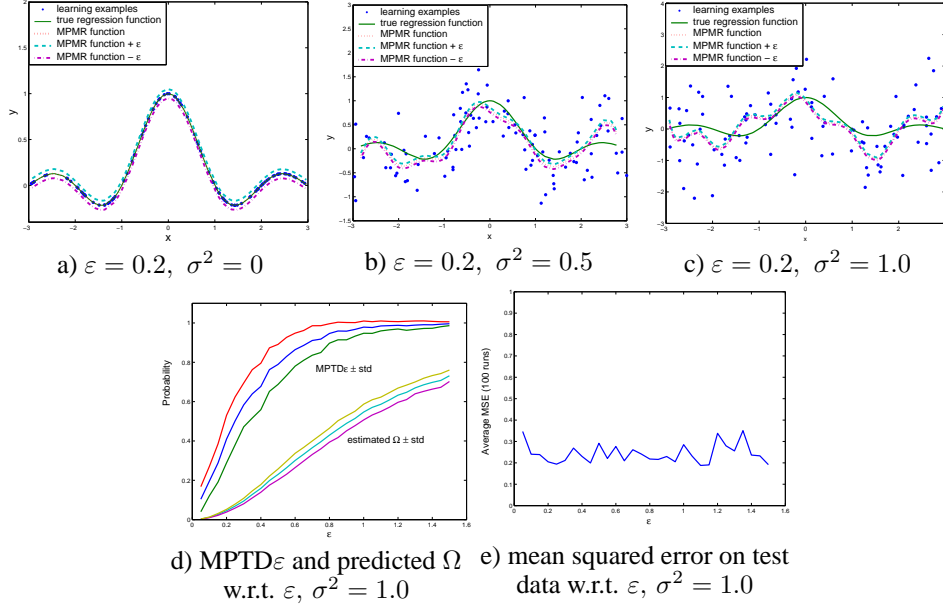

a) $\varepsilon = 0.2,\ \sigma^2 = 0$  b) $\varepsilon = 0.2,\ \sigma^2 = 0.5$  c) $\varepsilon = 0.2,\ \sigma^2 = 1.0$

d) MPTD$\varepsilon$ and predicted $\Omega$  e) mean squared error on test
   w.r.t. $\varepsilon$, $\sigma^2 = 1.0$       data w.r.t. $\varepsilon$, $\sigma^2 = 1.0$

Figure 1: Experimental results on toy sinc data.

Table 2: Results over 100 random trials for the Boston Housing Data for $\varepsilon = 0.1, 1.0, 2.0, ..., 10.0$: mean squared errors and the standard deviation; MPDT$\varepsilon$: fraction of test points that are within $\epsilon$ of $y$ and the standard deviation; predicted $\Omega$: predicted probability that the model is within $\varepsilon$ of $y$ and standard deviation.

| $\varepsilon$ | 0.1 | 1.0 | 2.0 | 3.0 | 4.0 | 4.0 | 6.0 | 7.0 | 8.0 | 9.0 | 10.0 |
|---|---|---|---|---|---|---|---|---|---|---|---|
| MSE | 9.9 | 10.5 | 10.9 | 9.5 | 10.3 | 9.9 | 10.5 | 10.5 | 9.2 | 10.1 | 10.6 |
| STD | 5.9 | 9.5 | 8.6 | 5.9 | 8.1 | 8.0 | 8.5 | 8.1 | 5.3 | 6.9 | 7.6 |
| MPDT$\varepsilon$ | 0.05 | 0.33 | 0.58 | 0.76 | 0.84 | 0.89 | 0.93 | 0.95 | 0.97 | 0.97 | 0.98 |
| STD | 0.04 | 0.09 | 0.09 | 0.08 | 0.07 | 0.06 | 0.05 | 0.04 | 0.03 | 0.03 | 0.02 |
| $\Omega$ | 0.002 | 0.19 | 0.51 | 0.69 | 0.80 | 0.87 | 0.90 | 0.92 | 0.94 | 0.95 | 0.96 |
| STD | 0.0005 | 0.03 | 0.06 | 0.05 | 0.04 | 0.03 | 0.01 | 0.01 | 0.009 | 0.009 | 0.008 |

no attempt was made to optimize for $\sigma$. Second, as with the toy data, the errors are relatively independent of $\varepsilon$. Finally, we note that the mean predicted $\Omega$ is lower than the measured average MPTD$\varepsilon$, thus validating the the MPMR algorithm does indeed predict an effective lower bound on the probability that the regression model is within $\varepsilon$ of the true regression function.

## 4 Discussion and Conclusion

We formalize the regression problem as one of maximizing the minimum probability, $\Omega$, that the regression model is within $\pm\varepsilon$ of the true regression function. By estimating mean and covariance matrix statistics of the regression data (and making no other assumptions on the underlying true regression function distributions), the proposed minimax probability machine regression (MPMR) algorithm obtains a *direct* estimate of $\Omega$. Two theorems are presented proving that, given perfect knowledge of the mean and covariance statistics of the true regression function, the proposed MPMR algorithm directly computes the exact lower probability bound $\Omega$. We are unaware of any other nonlinear regression model formulation that has this property.

Experimental results are given showing: 1) the regression models produced are competitive with existing state of the art models; 2) the mean squared error on test data is relatively independent of the choice of $\varepsilon$; and 3) estimating mean and covariance statistics directly from the learning data gives accurate lower probability bound $\Omega$ estimates that the regression model is within $\pm\varepsilon$ of the true regression function - thus supporting our theoretical results.

Future research will focus on a theoretical analysis of the conditions under which the accuracy of the regression model is independent of $\varepsilon$. Also, we are analyzing the rate, as a function of sample size, at which estimates of the lower probability bound $\Omega$ converge to the true value. Finally, the proposed minimax probability machine regression framework is a new formulation of the regression problem, and therefore its properties can only be fully understood through extensive experimentation. We are currently applying MPMR to a wide variety of regression problems and have made Matlab / C source code available (*http://www.cs.colorado.edu/~grudic/software*) for others to do the same.

## References

[1] G. R. G. Lanckriet, L. E. Ghaoui, C. Bhattacharyya, and M. I. Jordan. Minimax probability machine. In T. G. Dietterich, S. Becker, and Z. Ghahramani, editors, *Advances in Neural Information Processing Systems 14*, Cambridge, MA, 2002. MIT Press.

[2] A. W. Marshall and I. Olkin. Multivariate chebyshev inequalities. *Annals of Mathematical Statistics*, 31(4):1001–1014, 1960.

[3] I. Popescu and D. Bertsimas. Optimal inequalities in probability theory: A convex optimization approach. Technical Report TM62, INSEAD, Dept. Math. O.R., Cambridge, Mass, 2001.

[4] W. H. Press, B. P. Flannery, S. A. Teukolsky, and W. T. Vetterling. *Numerical Recipes in C*. Cambridge University Press, New York NY, 1988.

[5] Bernhard Schölkopf, Peter L. Bartlett, Alex J. Smola, and Robert Williamson. Shrinking the tube: A new support vector regression algorithm. In D. A. Cohn M. S. Kearns, S. A. Solla, editor, *Advances in Neural Information Processing Systems*, volume 11, Cambridge, MA, 1999. The MIT Press.

**Appendix: Proofs of Theorems 1 and 2**

**Proof of Theorem 1:**

Consider any point $\mathbf{x} = (x_1, ..., x_d)$ generated according to the distribution $\Lambda$. This point will have a corresponding $y$ (defined in (2)), and from (10), the probability that $\mathbf{z}_{+\varepsilon} = (y + \varepsilon, x_1, ..., x_d)$ will be classified correctly (as belonging to class $\mathbf{u}$) by (6) is $\alpha$. Furthermore, the classification boundary occurs uniquely at the point where $\mathbf{z} = (\hat{y}, x_1, ..., x_d)$, where, from the assumptions, $\hat{y}$ is the unique solution to (12). Similarly, for the same point $y$, the probability that $\mathbf{z}_{-\varepsilon} = (y - \varepsilon, x_1, ..., x_d)$ will be classified correctly (as belonging to class $\mathbf{v}$) by (6) is also $\alpha$, and the classifications boundary occurs uniquely at the point where $\mathbf{z} = (\hat{y}, x_1, ..., x_d)$. Therefore, both $\mathbf{z}_{+\varepsilon} = (y + \varepsilon, x_1, ..., x_d)$ and $\mathbf{z}_{-\varepsilon} = (y - \varepsilon, x_1, ..., x_d)$ are, with probability $\alpha$, on the correct side of the regression surface, defined by $\mathbf{z} = (\hat{y}, x_1, ..., x_d)$. Therefore, $\mathbf{z}_{+\varepsilon}$ differs from $\mathbf{z}$ by at most $+\varepsilon$ in the first dimension, and $\mathbf{z}_{-\varepsilon}$ differs from $\mathbf{z}$ by at most $-\varepsilon$ in the first dimension. Thus, the minimum bound on the probability that $|y - \hat{y}| \leq \varepsilon$ is $\alpha$ (defined in (10)), which has the same form as $\Omega$. This completes the proof. □

**Proof of Lemma 1:**

$[\tilde{k}_u]_i - [\tilde{k}_v]_i = \frac{1}{N}(\sum_{l=1}^{N} K^c(\mathbf{u_l}, \mathbf{z_i})) - \frac{1}{N}(\sum_{l=1}^{N} K^c(\mathbf{v_l}, \mathbf{z_i})) =$
$\frac{1}{N}\sum_{l=1}^{N}(y_l + \epsilon)y_i' + K(\mathbf{x_l}, \mathbf{x_i}) - ((y_l - \epsilon)y_i' + K(\mathbf{x_l}, \mathbf{x_i})) = \frac{1}{N}N2\epsilon y_i' = 2\epsilon y_i'$  $\square$

**Proof of Theorem 2:**

*Part 1:* Plugging (14) into (12), we get:

$$0 = \sum_{i=1}^{2N} \gamma_i \left[ y_i'\hat{y} + K(\mathbf{x}_i, \mathbf{x}) \right] + b_c$$

$$0 = \sum_{i=1}^{N} \gamma_i \left[ (y_i + \varepsilon)\hat{y} + K(\mathbf{x}_i, \mathbf{x}) \right] + \sum_{i=1}^{N} \gamma_{i+N} \left[ (y_i - \varepsilon)\hat{y} + K(\mathbf{x}_i, \mathbf{x}) \right] + b_c$$

$$0 = \sum_{i=1}^{N} \left\{ (\gamma_i + \gamma_{i+N}) \left[ y_i\hat{y} + K(\mathbf{x}_i, \mathbf{x}) \right] + (\gamma_i - \gamma_{i+N}) \varepsilon\hat{y} \right\} + b_c$$

When we solve analytically for $\hat{y}$, giving (5), the coefficients $\beta_i$ and the offset $b$ have a denominator that looks like: $-\sum_{i=1}^{N} \left[ (\gamma_i + \gamma_{i+N}) y_i + (\gamma_i - \gamma_{i+N}) \varepsilon \right] = -\gamma^{\mathbf{T}}\mathbf{y}'$

Applying Lemma 1 and (7) we obtain: $1 = \gamma^{\mathbf{T}}((\tilde{k}_u) - \tilde{k}_v) = \gamma^{\mathbf{T}}2\epsilon\mathbf{y}' \Leftrightarrow -\gamma^{\mathbf{T}}\mathbf{y}' = -\frac{1}{2\epsilon}$
for the denominator of $\beta_i$ and $b$.  $\square$

*Part 2:* The values $z_i$ are defined as: $z_1 = u_1, ..., z_N = u_N, z_{N+1} = v_1 = u_1 - (2\epsilon, 0, \cdots, 0)^T, ..., z_{2N} = v_N = u_N - (2\epsilon, 0, \cdots, 0)^T$. Since $\tilde{\mathbf{K}}_{\mathbf{u}} = \mathbf{K}_{\mathbf{u}} - 1_N\tilde{k}_{\mathbf{u}}$ we have the following term for a single matrix entry:

$[\tilde{K}_u]_{i,j} = K^c(\mathbf{u_i}, \mathbf{z_j}) - \frac{1}{N}\sum_{l=1}^{N} K^c(\mathbf{u_l}, \mathbf{z_j}) \quad i = 1, .., N \; j = 1, ..., 2N$

Similarly the matrix entries for $\tilde{K}_v$ look like:
$[\tilde{K}_v]_{i,j} = K^c(\mathbf{v_i}, \mathbf{z_j}) - \frac{1}{N}\sum_{l=1}^{N} K^c(\mathbf{v_l}, \mathbf{z_j}) \quad i = 1, .., N \; j = 1, ..., 2N$

We show that these entries are the same for all $i$ and $j$:
$[\tilde{K}_u]_{i,j} = K^c(\mathbf{v_i} + (2\epsilon\; 0\; \cdots\; 0)^T, \mathbf{z_j}) - \frac{1}{N}\sum_{l=1}^{N} K^c(\mathbf{v_l} + (2\epsilon\; 0\; \cdots\; 0)^T, \mathbf{z_j}) =$

$K^c(\mathbf{v_i}, \mathbf{z_j}) + 2\epsilon[z_j]_1 - \frac{1}{N}(\sum_{l=1}^{N} K^c(\mathbf{v_l}, \mathbf{z_j}) + 2\epsilon[z_j]_1) =$

$K^c(\mathbf{v_i}, \mathbf{z_j}) + 2\epsilon[z_j]_1 - \frac{1}{N}\sum_{l=1}^{N} K^c(\mathbf{v_l}, \mathbf{z_j}) - \frac{1}{N}\sum_{l=1}^{N} 2\epsilon[z_j]_1 =$

$K^c(\mathbf{v_i}, \mathbf{z_j}) + 2\epsilon[z_j]_1 - \frac{1}{N}\sum_{l=1}^{N} K^c(\mathbf{v_l}, \mathbf{z_j}) - \frac{1}{N}N2\epsilon[z_j]_1 =$

$K^c(\mathbf{v_i}, \mathbf{z_j}) - \frac{1}{N}\sum_{l=1}^{N} K^c(\mathbf{v_l}, \mathbf{z_j}) = [\tilde{K}_v]_{i,j}$

This completes the proof of *Part 2*.  $\square$

*Part 3:* From *Part 2* we know that $\tilde{\mathbf{K}}_{\mathbf{u}} = \tilde{\mathbf{K}}_{\mathbf{v}}$. Therefore, the minimization problem (7) collapses to $min\|\tilde{\mathbf{K}}_{\mathbf{u}}\gamma\|_2^2$ with respect to $\gamma$ (the N is constant and can be removed). Formulating this minimization with the use of the orthogonal matrix $\mathbf{F}$ and an initial vector $\gamma_{\mathbf{o}}$ this becomes (see [1]): $min\|\tilde{\mathbf{K}}_{\mathbf{u}}(\gamma_{\mathbf{o}} + \mathbf{Ft})\|_2^2$ with respect to $\mathbf{t} \in \Re^{2N-1}$. We set $h(\mathbf{t}) = \|\tilde{\mathbf{K}}_{\mathbf{u}}(\gamma + \mathbf{Ft})\|_2^2$. Therefore in order to find the minimum we must solve $2N - 1$ linear equations: $0 = \frac{d}{dt_i}h(\mathbf{t}) \quad i = 1, ..., 2N - 1$. This completes the proof of *Part 3*.  $\square$